# Bayesian Nonparametric Models on Decomposable Graphs

**François Caron**
INRIA Bordeaux Sud–Ouest
Institut de Mathématiques de Bordeaux
University of Bordeaux, France
`francois.caron@inria.fr`

**Arnaud Doucet**
Departments of Computer Science & Statistics
University of British Columbia, Vancouver, Canada
*and* The Institute of Statistical Mathematics
Tokyo, Japan
`arnaud@cs.ubc.ca`

## Abstract

Over recent years Dirichlet processes and the associated Chinese restaurant process (CRP) have found many applications in clustering while the Indian buffet process (IBP) is increasingly used to describe latent feature models. These models are attractive because they ensure exchangeability (over samples). We propose here extensions of these models where the dependency between samples is given by a known decomposable graph. These models have appealing properties and can be easily learned using Monte Carlo techniques.

## 1  Motivation

The CRP and IBP have found numerous applications in machine learning over recent years [5, 10]. We consider here the case where the data we are interested in are 'locally' dependent; these dependencies being represented by a **known** graph $\mathcal{G}$ where each data point/object is associated to a vertex. These local dependencies can correspond to any conceptual or real (e.g. space, time) metric. For example, in the context of clustering, we might want to propose a prior distribution on partitions enforcing that data which are 'close' in the graph are more likely to be in the same cluster. Similarly, in the context of latent feature models, we might be interested in a prior distribution on features enforcing that data which are 'close' in the graph are more likely to possess similar features.

The 'standard' CRP and IBP correspond to the case where the graph $\mathcal{G}$ is complete; that is it is fully connected. In this paper, we generalize the CRP and IBP to decomposable graphs. The resulting generalized versions of the CRP and IBP enjoy attractive properties. Each clique of the graph follows marginally a CRP or an IBP process and explicit expressions for the joint prior distribution on the graph is available. It makes it easy to learn those models using straightforward generalizations of Markov chain Monte Carlo (MCMC) or Sequential Monte Carlo (SMC) algorithms proposed to perform inference for the CRP and IBP [5, 10, 14].

The rest of the paper is organized as follows. In Section 2, we review the popular Dirichlet multinomial allocation model and the Dirichlet Process (DP) partition distribution. We propose an extension of these two models to decomposable graphical models. In Section 3 we discuss nonparametric latent feature models, reviewing briefly the construction in [5] and extending it to decomposable graphs. We demonstrate these models in Section 4 on two applications: an alternative to the hierarchical DP model [12] and a time-varying matrix factorization problem.

## 2  Prior distributions for partitions on decomposable graphs

Assume we have $n$ observations. When performing clustering, we associate to each of this observation an *allocation variable* $z_i \in [K] = \{1, \dots, K\}$. Let $\Pi_n$ be the partition of $[n] = \{1, \dots, n\}$ defined by the equivalence relation $i \leftrightarrow j \Leftrightarrow z_i = z_j$. The resulting partition $\Pi_n = \{A_1, \dots, A_{n(\Pi_n)}\}$

is an unordered collection of disjoint non-empty subsets $A_j$ of $[n]$, $j = 1, \ldots, n(\Pi_n)$, where $\cup_j A_j = [n]$ and $n(\Pi_n)$ is the number of subsets for partition $\Pi_n$. We also denote by $\mathcal{P}_n$ be the set of all partitions of $[n]$ and let $n_j$, $j = 1, \ldots, n(\Pi_n)$, be the size of the subset $A_j$.

Each allocation variable $z_i$ is associated to a vertex/site of an undirected graph $\mathcal{G}$, which is assumed to be known. In the standard case where the graph $\mathcal{G}$ is complete, we first review briefly here two popular prior distributions on $z_{1:n}$, equivalently on $\Pi_n$. We then extend these models to undirected decomposable graphs; see [2, 8] for an introduction to decomposable graphs. Finally we briefly discuss the directed case. Note that the models proposed here are completely different from the hyper multinomial-Dirichlet in [2] and its recent DP extension [6].

## 2.1 Dirichlet multinomial allocation model and DP partition distribution

Assume for the time being that $K$ is finite. When the graph is complete, a popular choice for the allocation variables is to consider a Dirichlet multinomial allocation model [11]

$$\pi \sim \mathcal{D}(\frac{\theta}{K}, \ldots, \frac{\theta}{K}), \; z_i|\pi \sim \pi \tag{1}$$

where $\mathcal{D}$ is the standard Dirichlet distribution and $\theta > 0$. Integrating out $\pi$, we obtain the following Dirichlet multinomial prior distribution

$$\Pr(z_{1:n}) = \frac{\Gamma(\theta) \prod_{j=1}^{K} \Gamma(n_j + \frac{\theta}{K})}{\Gamma(\theta + n)\Gamma(\frac{\theta}{K})^K} \tag{2}$$

and then, using the straightforward equality $\Pr(\Pi_n) = \frac{K!}{(K-n(\Pi_n))!} \Pr(z_{1:n})$ valid for for all $\Pi_n \in \mathcal{P}_K$ where $\mathcal{P}_K = \{\Pi_n \in \mathcal{P}_n | n(\Pi_n) \le K\}$, we obtain

$$\Pr(\Pi_n) = \frac{K!}{(K-n(\Pi_n))!} \frac{\Gamma(\theta) \prod_{j=1}^{n(\Pi_n)} \Gamma(n_j + \frac{\theta}{K})}{\Gamma(\theta + n)\Gamma(\frac{\theta}{K})^{n(\Pi_n)}}. \tag{3}$$

DP may be seen as a generalization of the Dirichlet multinomial model when the number of components $K \to \infty$; see for example [10]. In this case the distribution over the partition $\Pi_n$ of $[n]$ is given by [11]

$$\Pr(\Pi_n) = \frac{\theta^{n(\Pi_n)} \prod_{j=1}^{n(\Pi_n)} \Gamma(n_j)}{\prod_{i=1}^{n}(\theta + i - 1)}. \tag{4}$$

Let $\Pi_{-k} = \{A_{1,-k}, \ldots, A_{n(\Pi_{-k}),-k}\}$ be the partition induced by removing item $k$ to $\Pi_n$ and $n_{j,-k}$ be the size of cluster $j$ for $j = 1, \ldots, n(\Pi_{-k})$. It follows from (4) that an item $k$ is assigned to an existing cluster $j$, $j = 1, \ldots, n(\Pi_{-k})$, with probability proportional to $n_{j,-k}/(n - 1 + \theta)$ and forms a new cluster with probability $\theta/(n - 1 + \theta)$. This property is the basis of the CRP. We now extend the Dirichlet multinomial allocation and the DP partition distribution models to decomposable graphs.

## 2.2 Markov combination of Dirichlet multinomial and DP partition distributions

Let $\mathcal{G}$ be a decomposable undirected graph, $\mathcal{C} = \{C_1, \ldots, C_p\}$ a perfect ordering of the cliques and $\mathcal{S} = \{S_2, \ldots, C_p\}$ the associated separators. It can be easily checked that if the marginal distribution of $z_C$ for each clique $C \in \mathcal{C}$ is defined by (2) then these distributions are consistent as they yield the same distribution (2) over the separators. Therefore, the unique Markov distribution over $\mathcal{G}$ with Dirichlet multinomial distribution over the cliques is defined by [8]

$$\Pr(z_{1:n}) = \frac{\prod_{C \in \mathcal{C}} \Pr(z_C)}{\prod_{S \in \mathcal{S}} \Pr(z_S)} \tag{5}$$

where for each complete set $B \subseteq \mathcal{G}$, we have $\Pr(z_B)$ given by (2). It follows that we have for any $\Pi_n \in \mathcal{P}_K$

$$\Pr(\Pi_n) = \frac{K!}{(K-n(\Pi_n))!} \frac{\prod_{C \in \mathcal{C}} \frac{\Gamma(\theta) \prod_{j=1}^{K} \Gamma(n_{j,C} + \frac{\theta}{K})}{\Gamma(\theta+n_C)\Gamma(\frac{\theta}{K})^K}}{\prod_{S \in \mathcal{S}} \frac{\Gamma(\theta) \prod_{j=1}^{K} \Gamma(n_{j,S} + \frac{\theta}{K})}{\Gamma(\theta+n_S)\Gamma(\frac{\theta}{K})^K}} \tag{6}$$

where for each complete set $B \subseteq \mathcal{G}$, $n_{j,B}$ is the number of items associated to cluster $j$, $j = 1, \ldots, K$ in $B$ and $n_B$ is the total number of items in $B$. Within each complete set $B$, the allocation variables define a partition distributed according to the Dirichlet-multinomial distribution.

We now extend this approach to DP partition distributions; that is we derive a joint distribution over $\Pi_n$ such that the distribution of $\Pi_B$ over each complete set $B$ of the graph is given by (4) with $\theta > 0$. Such a distribution satisfies the consistency condition over the separators as the restriction of any partition distributed according to (4) still follows (4) [7].

**Proposition**. Let $\mathcal{P}_n^{\mathcal{G}}$ be the set of partitions $\Pi_n \in \mathcal{P}_n$ such that for each decomposition $A$, $B$, and any $(i,j) \in A \times B$, $i \leftrightarrow j \Rightarrow \exists k \in A \cap B$ such that $k \leftrightarrow i \leftrightarrow j$. As $K \to \infty$, the prior distribution over partitions (6) is given for each $\Pi_n \in \mathcal{P}_n^{\mathcal{G}}$ by

$$\Pr(\Pi_n) = \theta^{n(\Pi_n)} \frac{\prod_{C \in \mathcal{C}} \frac{\prod_{j=1}^{n(\Pi_C)} \Gamma(n_{j,C})}{\prod_{i=1}^{n_C}(\theta+i-1)}}{\prod_{S \in \mathcal{S}} \frac{\prod_{j=1}^{n(\Pi_S)} \Gamma(n_{j,S})}{\prod_{i=1}^{n_S}(\theta+i-1)}} \tag{7}$$

where $n(\Pi_B)$ is the number of clusters in the complete set $B$.

**Proof**. From (6), we have

$$\Pr(\Pi_n) = \frac{K(K-1)\ldots(K-n(\Pi_n)+1)}{K^{\sum_{C \in \mathcal{C}} n(\Pi_C) - \sum_{S \in \mathcal{S}} n(\Pi_S)}} \frac{\prod_{C \in \mathcal{C}} \frac{\theta^{n(\Pi_C)} \prod_{j=1}^{n(\Pi_C)} \Gamma(n_{j,C}+\frac{\theta}{K})}{\prod_{i=1}^{n_C}(\theta+i-1)}}{\prod_{S \in \mathcal{S}} \frac{\theta^{n(\Pi_S)} \prod_{j=1}^{n(\Pi_S)} \Gamma(n_{j,S}+\frac{\theta}{K})}{\prod_{i=1}^{n_S}(\theta+i-1)}}$$

Thus when $K \to \infty$, we obtain (7) if $n(\Pi_n) = \sum_{C \in \mathcal{C}} n(\Pi_C) - \sum_{S \in \mathcal{S}} n(\Pi_S)$ and 0 otherwise. We have $n(\Pi_n) \leq \sum_{C \in \mathcal{C}} n(\Pi_C) - \sum_{S \in \mathcal{S}} n(\Pi_S)$ for any $\Pi_n \in \mathcal{P}_n$ and the subset of $\mathcal{P}_n$ verifying $n(\Pi_n) = \sum_{C \in \mathcal{C}} n(\Pi_C) - \sum_{S \in \mathcal{S}} n(\Pi_S)$ corresponds to the set $\mathcal{P}_n^{\mathcal{G}}$.∎

**Example**. Let the notation $i \sim j$ (resp. $i \nsim j$) indicates an edge (resp. no edge) between two sites. Let $n = 3$ and $\mathcal{G}$ be the decomposable graph defined by the relations $1 \sim 2$, $2 \sim 3$ and $1 \nsim 3$. The set $\mathcal{P}_3^{\mathcal{G}}$ is then equal to $\{\{\{1,2,3\}\}; \{\{1,2\}, \{3\}\}; \{\{1\}, \{2,3\}\}; \{\{1\}, \{2\}, \{3\}\}\}$. Note that the partition $\{\{1,3\}, \{2\}\}$ does not belong to $\mathcal{P}_3^{\mathcal{G}}$. Indeed, as there is no edge between 1 and 3, they cannot be in the same cluster if 2 is in another cluster. The cliques are $C_1 = \{1,2\}$ and $C_2 = \{2,3\}$ and the separator is $S_2 = \{2\}$. The distribution is given by $\Pr(\Pi_3) = \frac{\Pr(\Pi_{C_1})\Pr(\Pi_{C_2})}{\Pr(\Pi_{S_2})}$ hence we can check that we obtain $\Pr(\{1,2,3\}) = (\theta+1)^{-2}$, $\Pr(\{1,2\}, \{3\}) = \Pr(\{1,2\}, \{3\}) = \theta(\theta+1)^{-2}$ and $\Pr(\{1\}, \{2\}, \{3\}) = \theta^2(\theta+1)^{-2}$.∎

Let now define the full conditional distributions. Based on (7) the conditional assignment of an item $k$ is proportional to the conditional over the cliques divided by the conditional over the separators. Let denote $\mathcal{G}_{-k}$ the undirected graph obtained by removing vertex $k$ from $\mathcal{G}$. Suppose that $\Pi_n \in \mathcal{P}_n^{\mathcal{G}}$. If $\Pi_{-k} \notin \mathcal{P}_{n-1}^{\mathcal{G}_{-k}}$, then do not change the value of item $k$. Otherwise, item $k$ is assigned to cluster $j$ where $j = 1, \ldots, n(\Pi_{-k})$ with probability proportional to

$$\frac{\prod_{\{C \in \mathcal{C} | n_{-k,j,C} > 0\}} n_{-k,j,C}}{\prod_{\{S \in \mathcal{S} | n_{-k,j,S} > 0\}} n_{-k,j,S}} \tag{8}$$

and to a new cluster with probability proportional to $\theta$, where $n_{-k,j,C}$ is the number of items in the set $C \setminus \{k\}$ belonging to cluster $j$. The updating process is illustrated by the *Chinese wedding party process*[1] in Fig. 1. The results of this section can be extended to the Pitman-Yor process, and more generally to species sampling models.

**Example** (continuing). Given $\Pi_{-2} = \{A_1 = \{1\}, A_2 = \{3\}\}$, we have $\Pr(\text{item } 2 \text{ assigned to } A_1 = \{1\} | \Pi_{-2}) = \Pr(\text{item } 2 \text{ assigned to } A_2 = \{3\} | \Pi_{-2}) = (\theta+2)^{-1}$ and $\Pr(\text{item } 2 \text{ assigned to new cluster } A_3 | \Pi_{-2}) = \theta(\theta+2)^{-1}$. Given $\Pi_{-2} = \{A_1 = \{1,3\}\}$, item 2 is assigned to $A_1$ with probability 1.∎

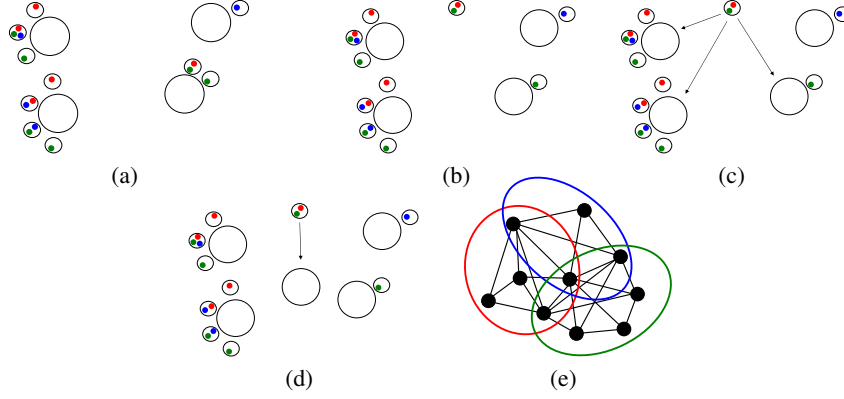

Figure 1: Chinese wedding party. Consider a group of $n$ guests attending a wedding party. Each of the $n$ guests may belong to one or several cliques, i.e. maximal groups of people such that everybody knows everybody. The belonging of each guest to the different cliques is represented by color patches on the figures, and the graphical representation of the relationship between the guests is represented by the graphical model (e). (a) Suppose that the guests are already seated such that two guests cannot be together at the same table is they are not part of the same clique, or if there does not exist a group of other guests such that they are related ("*Any friend of yours is a friend of mine*"). (b) The guest number $k$ leaves his table and either (c) joins a table where there are guests from the same clique as him, with probability proportional to the product of the number of guests from each clique over the product of the number of guests belonging to several cliques on that table or (d) he joins a new table with probability proportional to $\theta$.

## 2.3 Monte Carlo inference

### 2.3.1 MCMC algorithm

Using the full conditionals, a single site Gibbs sampler can easily be designed to approximate the posterior distribution $\Pr(\Pi_n|\mathbf{z}_{1:n})$. Given a partition $\Pi_n$, an item $k$ is taken out of the partition. If $\Pi_{-k} \notin \mathcal{P}_{n-1}^{\mathcal{G}-k}$, item $k$ keeps the same value. Otherwise, the item will be assigned to a cluster $j$, $j = 1, \ldots, n(\Pi_{-k})$, with probability proportional to

$$\frac{p(\mathbf{z}_{\{k\}\cup A_{j,-k}})}{p(\mathbf{z}_{A_{j,-k}})} \times \frac{\prod_{\{C\in\mathcal{C}|n_{-k,j,C}>0\}} n_{-k,j,C}}{\prod_{\{S\in\mathcal{S}|n_{-k,j,S}>0\}} n_{-k,j,S}} \qquad (9)$$

and the item will be assigned to a new cluster with probability proportional to $p(\mathbf{z}_{\{k\}}) \times \theta$. Similarly to [3], we can also define a procedure to sample from $p(\theta|n(\Pi_n) = k))$. We assume that $\theta \sim \mathcal{G}(a,b)$ and use $p$ auxiliary variables $x_1, \ldots, x_p$. The procedure is as follows.

- For $j = 1, \ldots, p$, sample $x_j|k, \theta \sim Beta(\theta + n_{S_j}, n_{C_j} - n_{S_j})$
- Sample $\theta|k, x_{1:p} \sim \mathcal{G}(a + k, b - \sum_j \log x_j)$

### 2.3.2 Sequential Monte Carlo

We have so far only treated the case of an undirected decomposable graph $\mathcal{G}$. We can formulate a sequential updating rule for the corresponding perfect directed version $\mathcal{D}$ of $\mathcal{G}$. Indeed, let $(a_1, \ldots a_{|V|})$ be a perfect ordering and $pa(a_k)$ be the set of parents of $a_k$ which is by definition complete. Let $\Pi_{k-1} = \{A_{1,k-1}, \ldots, A_{n(\Pi_{k-1}),k-1}\}$ denote the partition of the first $k-1$ vertices $a_{1:k-1}$ and let $n_{j,pa(a_k)}$ be the number of elements with value $j$ in the set $pa(a_k)$, $j = 1, \ldots, n(\Pi_{k-1})$. Then the vertex $a_k$ joins the set $j$ with probability $n_{j,pa(a_k)}/\left(\theta + \sum_q n_{q,pa(a_k)}\right)$ and creates a new cluster with probability $\theta/\left(\theta + \sum_q n_{q,pa(a_k)}\right)$.

One can then design a particle filter/SMC method in a similar fashion as [4]. Consider a set of $N$ particles $\Pi_{k-1}^{(i)}$ with weights $w_{k-1}^{(i)} \propto \Pr(\Pi_{k-1}^{(i)}, \mathbf{z}_{1:k-1})$ ($\sum_{i=1}^N w_{k-1}^{(i)} = 1$) that approximate the posterior distribution $\Pr(\Pi_{k-1}|\mathbf{z}_{1:k-1})$. For each particle $i$, there are $n(\Pi_{k-1}^{(i)}) + 1$ possible

allocations for component $a_k$. We denote $\widetilde{\Pi}_k^{(i,j)}$ the partition obtained by associating component $a_k$ to cluster $j$. The weight associated to $\widetilde{\Pi}_k^{(i,j)}$ is given by

$$\widetilde{w}_{k-1}^{(i,j)} = w_{k-1}^{(i)} \frac{p(\mathbf{z}_{\{a_k\} \cup A_{j,k-1}})}{p(\mathbf{z}_{A_{j,k-1}})} \times \begin{cases} \frac{n_{j,pa(a_k)}}{\theta + \sum_q n_{q,pa(a_k)}} & \text{if } j = 1, \dots, n(\Pi_{k-1}^{(i)}) \\ \frac{\theta}{\theta + \sum_q n_{q,pa(a_k)}} & \text{if } j = n(\Pi_{k-1}^{(i)}) + 1 \end{cases} \tag{10}$$

Then we can perform a deterministic resampling step by keeping the $N$ particles $\widetilde{\Pi}_k^{(i,j)}$ with highest weights $\widetilde{w}_{k-1}^{(i,j)}$. Let $\Pi_k^{(i)}$ be the resampled particles and $w_k^{(i)}$ the associated normalized weights.

## 3 Prior distributions for infinite binary matrices on decomposable graphs

Assume we have $n$ objects; each of these objects being associated to the vertex of a graph $\mathcal{G}$. To each object is associated a $K$-dimensional binary vector $\mathbf{z}_n = (z_{n,1}, \dots, z_{n,K}) \in \{0,1\}^K$ where $z_{n,i} = 1$ if object $n$ possesses feature $i$ and $z_{n,i} = 0$ otherwise. These vectors $\mathbf{z}_t$ form a binary $n \times K$ matrix denoted $\mathbf{Z}_{1:n}$. We denote by $\xi_{1:n}$ the associated equivalence class of left-ordered matrices and let $\mathcal{E}_K$ be the set of left-ordered matrices with at most $K$ features.

In the standard case where the graph $\mathcal{G}$ is complete, we review briefly here two popular prior distributions on $\mathbf{Z}_{1:n}$, equivalently on $\xi_{1:n}$: the Beta-Bernoulli model and the IBP [5]. We then extend these models to undirected decomposable graphs. This can be used for example to define a time-varying IBP as illustrated in Section 4.

### 3.1 Beta-Bernoulli and IBP distributions

The Beta-Bernoulli distribution over the allocation $\mathbf{Z}_{1:n}$ is

$$\Pr(\mathbf{Z}_{1:n}) = \prod_{j=1}^{K} \frac{\frac{\alpha}{K}\Gamma(n_j + \frac{\alpha}{K})\Gamma(n - n_j + 1)}{\Gamma(n + 1 + \frac{\alpha}{K})} \tag{11}$$

where $n_j$ is the number of objects having feature $j$. It follows that

$$\Pr(\xi_{1:n}) = \frac{K!}{\prod_{h=0}^{2^n-1} K_h!} \prod_{j=1}^{K} \frac{\frac{\alpha}{K}\Gamma(n_j + \frac{\alpha}{K})\Gamma(n - n_j + 1)}{\Gamma(n + 1 + \frac{\alpha}{K})} \tag{12}$$

where $K_h$ is the number of features possessing the history $h$ (see [5] for details). The nonparametric model is obtained by taking the limit when $K \to \infty$

$$\Pr(\xi_{1:n}) = \frac{\alpha^{K^+}}{\prod_{h=1}^{2^n-1} K_h!} \exp(-\alpha H_n) \prod_{j=1}^{K^+} \frac{(n - n_j)!(n_j - 1)!}{n!} \tag{13}$$

where $K^+$ is the total number of features and $H_n = \sum_{k=1}^{n} \frac{1}{k}$. The IBP follows from (13).

### 3.2 Markov combination of Beta-Bernoulli and IBP distributions

Let $\mathcal{G}$ be a decomposable undirected graph, $\mathcal{C} = \{C_1, \dots, C_p\}$ a perfect ordering of the cliques and $\mathcal{S} = \{S_2, \dots, C_p\}$ the associated separators. As in the Dirichlet-multinomial case, it is easily seen that if for each clique $C \in \mathcal{C}$, the marginal distribution is defined by (11), then these distributions are consistent as they yield the same distribution (11) over the separators. Therefore, the unique Markov distribution over $\mathcal{G}$ with Beta-Bernoulli distribution over the cliques is defined by [8]

$$\Pr(\mathbf{Z}_{1:n}) = \frac{\prod_{C \in \mathcal{C}} \Pr(\mathbf{Z}_C)}{\prod_{S \in \mathcal{S}} \Pr(\mathbf{Z}_S)} \tag{14}$$

where $\Pr(\mathbf{Z}_B)$ given by (11) for each complete set $B \subseteq \mathcal{G}$. The prior over $\xi_{1:n}$ is thus given, for $\xi_{1:n} \in \mathcal{E}_K$, by

$$\Pr(\xi_{1:n}) = \frac{K!}{\prod_{h=0}^{2^n-1} K_h!} \frac{\prod_{C \in \mathcal{C}} \prod_{j=1}^{K} \frac{\frac{\alpha}{K}\Gamma(n_{j,C} + \frac{\alpha}{K})\Gamma(n_C - n_{j,C} + 1)}{\Gamma(n_C + 1 + \frac{\alpha}{K})}}{\prod_{S \in \mathcal{S}} \prod_{j=1}^{K} \frac{\frac{\alpha}{K}\Gamma(n_{j,S} + \frac{\alpha}{K})\Gamma(n_S - n_{j,S} + 1)}{\Gamma(n_S + 1 + \frac{\alpha}{K})}} \tag{15}$$

where for each complete set $B \subseteq \mathcal{G}$, $n_{j,B}$ is the number of items having feature $j$, $j = 1, \ldots, K$ in the set $B$ and $n_B$ is the whole set of objects in set $B$. Taking the limit when $K \to \infty$, we obtain after a few calculations

$$\Pr(\xi_{1:n}) = \frac{\alpha^{K^+_{[n]}} \exp\left[-\alpha\left(\sum_C H_{n_C} - \sum_S H_{n_S}\right)\right]}{\prod_{h=1}^{2^n-1} K_h!} \times \frac{\prod_{C \in \mathcal{C}} \prod_{j=1}^{K^+_C} \frac{(n_C - n_{j,C})!(n_{j,C}-1)!}{n_C!}}{\prod_{S \in \mathcal{S}} \prod_{j=1}^{K^+_S} \frac{(n_S - n_{j,S})!(n_{j,S}-1)!}{n_S!}}$$

if $K^+_{[n]} = \sum_C K^+_C - \sum_S K^+_S$ and 0 otherwise, where $K^+_B$ is the number of different features possessed by objects in $B$.

Let $\mathcal{E}^{\mathcal{G}}_n$ be the subset of $\mathcal{E}_n$ such that for each decomposition $A, B$ and any $(u, v) \in A \times B$: $\{u$ and $v$ possess feature $j\} \Rightarrow \exists k \in A \cap B$ such that $\{k$ possesses feature $j\}$. Let $\xi_{-k}$ be the left-ordered matrix obtained by removing object $k$ from $\xi_n$ and $K^+_{-k}$ be the total number of different features in $\xi_{-k}$. For each feature $j = 1, \ldots, K^+_{-k}$, if $\xi_{-k} \in \mathcal{E}^{\mathcal{G}_{-k}}_{n-1}$ then we have

$$\Pr(\xi_{k,j} = i) = \begin{cases} b \frac{\prod_{C \in \mathcal{C}} n_{j,C}}{\prod_{S \in \mathcal{C}} n_{j,S}} & \text{if } i = 1 \\ b \frac{\prod_{C \in \mathcal{C}}(n_C - n_{j,C})}{\prod_{S \in \mathcal{C}}(n_S - n_{j,S})} & \text{if } i = 0 \end{cases} \tag{16}$$

where $b$ is the appropriate normalizing constant then the customer $k$ tries Poisson$\left(\alpha \frac{\prod_{\{S \in \mathcal{S} | k \in S\}} n_S}{\prod_{\{C \in \mathcal{C} | k \in C\}} n_C}\right)$ new dishes. We can easily generalize this construction to a directed version $\mathcal{D}$ of $\mathcal{G}$ using arguments similar to those presented in Section 2; see Section 4 for an application to time-varying matrix factorization.

# 4 Applications

## 4.1 Sharing clusters among relative groups: An alternative to HDP

Consider that we are given $d$ groups with $n_j$ data $y_{i,j}$ in each group, $i = 1, \ldots, n_j$, $j = 1, \ldots, d$. We consider latent cluster variables $z_{i,j}$ that define the partition of the data. We will use alternatively the notation $\theta_{i,j} = U_{z_{i,j}}$ in the following. Hierarchical Dirichlet Process [12] (HDP) is a very popular model for sharing clusters among related groups. It is based on a hierarchy of DPs

$$G_0 \sim DP(\gamma, H),$$
$$G_j | G_0 \sim DP(\alpha, G_0) \quad j = 1, \ldots d$$
$$\theta_{i,j} | G_j \sim G_j, \ y_{i,j} | \theta_{i,j} \sim f(\theta_{i,j}) \ i = 1, \ldots, n_j.$$

Under conjugacy assumptions, $G_0$, $G_j$ and $U$ can be integrated out and we can approximate the marginal posterior of $(z_{i,j})$ given $y = (y_{i,j})$ with Gibbs sampling using the Chinese restaurant franchise to sample from the full conditional $p(z_{i,j} | z_{-\{i,j\}}, y)$.

Using the graph formulation defined in Section 2, we propose an alternative to HDP. Let $\theta_{0,1}, \ldots, \theta_{0,N}$ be $N$ auxiliary variables belonging to what we call group 0. We define each clique $C_j$ ($j = 1, \ldots, d$) to be composed of elements from group $j$ and elements from group 0. This defines a decomposable graphical model whose separator is given by the elements of group 0. We can rewrite the model in a way quite similar to HDP

$$G_0 \sim DP(\alpha, H),$$
$$\theta_{0,i} | G_0 \sim G_0 \quad i = 1, \ldots, N$$
$$G_j | \theta_{0,1}, \ldots, \theta_{0,N} \sim DP(\alpha + N, \frac{\alpha}{\alpha+N} H + \frac{\alpha}{\alpha+N} \sum_{i=1}^{N} \delta_{\theta_{0,i}}) \quad j = 1, \ldots d,$$
$$\theta_{i,j} | G_j \sim G_j, \ y_{i,j} | \theta_{i,j} \sim f(\theta_{i,j}) \ i = 1, \ldots, n_j$$

For any subset $A$ and $j \neq k \in \{1, \ldots, p\}$ we have $corr(G_j(A), G_k(A)) = \frac{N}{\alpha+N}$. Again, under conjugacy conditions, we can integrate out $G_0$, $G_j$ and $U$ and approximate the marginal posterior distribution over the partition using the Chinese wedding party process defined in Section 2. Note that for latent variables $z_{i,j}$, $j = 1, \ldots, d$, associated to data, this is the usual CRP update. As in HDP, multiple layers can be added to the model. Figures 2 (a) and (b) resp. give the graphical DP alternative to HDP and 2-layer HDP.

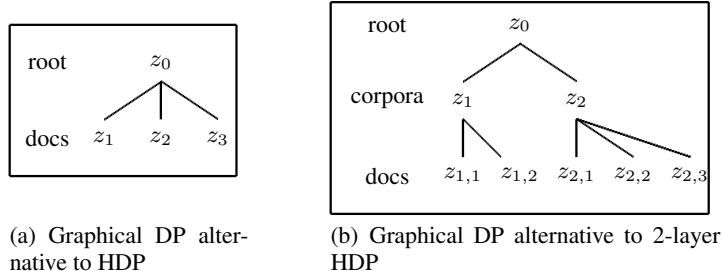

(a) Graphical DP alternative to HDP

(b) Graphical DP alternative to 2-layer HDP

Figure 2: Hierarchical Graphs of dependency with (a) one layer and (b) two layers of hierarchy.

If $N = 0$, then $G_j \sim DP(\alpha, H)$ for all $j$ and this is equivalent to setting $\gamma \to \infty$ in HDP. If $N \to \infty$ then $G_j = G_0$ for all $j$, $G_0 \sim DP(\alpha, H)$. This is equivalent to setting $\alpha \to \infty$ in the HDP. One interesting feature of the model is that, contrary to HDP, the marginal distribution of $G_j$ at any layer of the tree is $DP(\alpha, H)$. As a consequence, the total number of clusters scales logarithmically (as in the usual DP) with the size of each group, whereas it scales *doubly* logarithmically in HDP. Contrary to HDP, there are at most $N$ clusters shared between different groups. Our model is in that sense reminiscent of [9] where only a limited number of clusters can be shared. Note however that contrary to [9] we have a simple CRP-like process. The proposed methodology can be straightforwardly extended to the infinite HMM [12].

The main issue of the proposed model is the setting of the number $N$ of auxiliary parameters. Another issue is that to achieve high correlation, we need a large number of auxiliary variables. Nonetheless, the computational time used to sample from auxiliary variables is negligible compared to the time used for latent variables associated to data. Moreover, it can be easily parallelized. The model proposed offers a far richer framework and ensures that at each level of the tree, the marginal distribution of the partition is given by a DP partition model.

## 4.2 Time-varying matrix factorization

Let $\mathbf{X}_{1:n}$ be an observed matrix of dimension $n \times D$. We want to find a representation of this matrix in terms of two latent matrices $\mathbf{Z}_{1:n}$ of dimension $n \times K$ and $\mathbf{Y}$ of dimension $K \times D$. Here $\mathbf{Z}_{1:n}$ is a binary matrix whereas $\mathbf{Y}$ is a matrix of latent features. By assuming that $\mathbf{Y} \sim \mathcal{N}\left(0, \sigma_Y^2 I_{K \times D}\right)$ and

$$\mathbf{X}_{1:n} = \mathbf{Z}_{1:n}\mathbf{Y} + \sigma_X \varepsilon_n \text{ where } \varepsilon_n \sim \mathcal{N}\left(0, \sigma_X^2 I_{n \times D}\right),$$

we obtain

$$p(\mathbf{X}_{1:n}|\mathbf{Z}_{1:n}) \propto \frac{\left|\mathbf{Z}_{1:n}^{+\mathrm{T}}\mathbf{Z}_{1:n}^+ + \sigma_X^2/\sigma_Y^2 I_{K_n^+}\right|^{-D/2}}{\sigma_X^{(n-K_n^+)D}\sigma_Y^{K_n^+ D}} \exp\left\{-\frac{1}{2\sigma_X^2}\mathrm{tr}\left(\mathbf{X}_{1:n}^{\mathrm{T}}\Sigma_n^{-1}\mathbf{X}_{1:n}\right)\right\} \qquad (17)$$

where $\Sigma_n^{-1} = I - \mathbf{Z}_{1:n}^+\left(\mathbf{Z}_{1:n}^{+\mathrm{T}}\mathbf{Z}_{1:n}^+ + \sigma_X^2/\sigma_Y^2 I_{K_n^+}\right)^{-1}\mathbf{Z}_{1:n}^{+\mathrm{T}}$, $K_n^+$ the number of non-zero columns of $\mathbf{Z}_{1:n}$ and $\mathbf{Z}_{1:n}^+$ is the first $K_n^+$ columns of $\mathbf{Z}_{1:n}$. To avoid having to set $K$, [5, 14] assume that $\mathbf{Z}_{1:n}$ follows an IBP. The resulting posterior distribution $p(\mathbf{Z}_{1:n}|\mathbf{X}_{1:n})$ can be estimated through MCMC [5] or SMC [14].

We consider here a different model where the object $\mathbf{X}_t$ is assumed to arrive at time index $t$ and we want a prior distribution on $\mathbf{Z}_{1:n}$ ensuring that objects close in time are more likely to possess similar features. To achieve this, we consider the simple directed graphical model $\mathcal{D}$ of Fig. 3 where the site numbering corresponds to a time index in that case and a perfect numbering of $\mathcal{D}$ is $(1, 2, \ldots)$. The set of parents $pa(t)$ is composed of the $r$ preceding sites $\{\{t-r\}, \ldots, \{t-1\}\}$. The time-varying IBP to sample from $p(\mathbf{Z}_{1:n})$ associated to this directed graph follows from (16) and proceeds as follows.

At time $t = 1$
• Sample $K_1^{new} \sim \text{Poisson}(\alpha)$, set $z_{1,i} = 1$ for $i = 1, ..., K_1^{new}$ and set $K_1^+ = K_{new}$.

At times $t = 2, \ldots, r$
• For $k = 1, \ldots K_t^+$, sample $z_{t,k} \sim Ber\left(\frac{n_{1:t-1,k}}{t}\right)$ and $K_t^{new} \sim \text{Poisson}\left(\frac{\alpha}{t}\right)$.

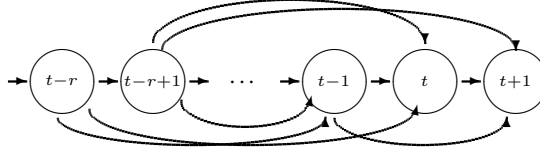

Figure 3: Directed graph.

At times $t = r+1, \ldots, n$
• For $k = 1, \ldots K_t^+$, sample $z_{t,k} \sim Ber(\frac{n_{t-r:t-1,k}}{r+1})$ and $K_t^{new} \sim \text{Poisson}(\frac{\alpha}{r+1})$.

Here $K_t^+$ is the total number of features appearing from time $\max(1, t-r)$ to $t-1$ and $n_{t-r:t-1,k}$ the restriction of $n_{1:t-1}$ to the $r$ last customers. Using (17) and the prior distribution of $\mathbf{Z}_{1:n}$ which can be sampled using the time-varying IBP described above, we can easily design an SMC method to sample from $p(\mathbf{Z}_{1:n}|\mathbf{X}_{1:n})$. We do not detail it here. Note that contrary to [14], our algorithm does not require inverting a matrix whose dimension grows linearly with the size of the data but only a matrix of dimension $r \times r$. In order to illustrate the model and SMC algorithm, we create $200$ $6 \times 6$ images using a ground truth $\mathbf{Y}$ consisting of 4 different $6 \times 6$ latent images. The $200 \times 4$ binary matrix was generated from $\Pr(z_{t,k} = 1) = \pi_{t,k}$, where $\pi_t = (\begin{array}{cccc} .6 & .5 & 0 & 0 \end{array})$ if $t = 1, \ldots, 30$, $\pi_t = (\begin{array}{cccc} .4 & .8 & .4 & 0 \end{array})$ if $t = 31, \ldots, 50$ and $\pi_t = (\begin{array}{cccc} 0 & .3 & .6 & .6 \end{array})$ if $t = 51, \ldots, 200$. The order of the model is set to $r = 50$. The feature occurences $\mathbf{Z}_{1:n}$ and true features $\mathbf{Y}$ and their estimates are represented in Figure 4. Two spurious features are detected by the model (features 2 and 5 on Fig. 3(c)) but quickly discarded (Fig. 4(d)). The algorithm is able to correctly estimate the varying prior occurences of the features over time.

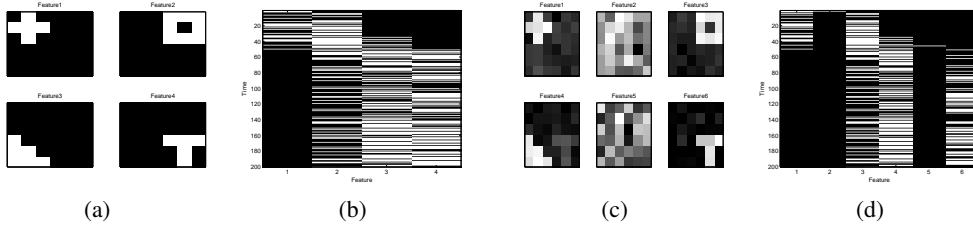

| (a) | (b) | (c) | (d) |

Figure 4: (a) True features, (b) True features occurences, (c) MAP estimate $\mathbf{Z}_{MAP}$ and (d) associated $E[\mathbf{Y}|\mathbf{Z}_{MAP}]$

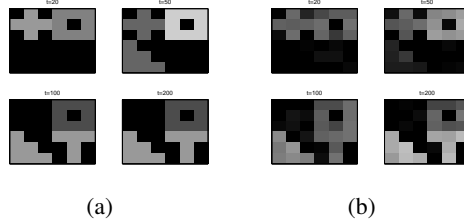

| (a) | (b) |

Figure 5: (a) $E[\mathbf{X}_t|\pi_t, \mathbf{Y}]$ and (b) $E[\mathbf{X}_t|\mathbf{X}_{1:t-1}]$ at $t = 20, 50, 100, 200$.

## 5 Related work and Discussion

The fixed-lag version of the time-varying DP of Caron et al. [1] is a special case of the proposed model when $\mathcal{G}$ is given by Fig. 3. The bivariate DP of Walker and Muliere [13] is also a special case when $\mathcal{G}$ has only two cliques. In this paper, we have assumed that the structure of the graph was known beforehand and we have shown that many flexible models arise from this framework. It would be interesting in the future to investigate the case where the graphical structure is unknown and must be estimated from the data.

## Acknowledgment

The authors thank the reviewers for their comments that helped to improve the writing of the paper.

## Footnotes

[1]Note that this representation describes the full conditionals while the CRP represents the sequential updating.

# References

[1] F. Caron, M. Davy, and A. Doucet. Generalized Polya urn for time-varying Dirichlet process mixtures. In *Uncertainty in Artificial Intelligence*, 2007.

[2] A.P. Dawid and S.L. Lauritzen. Hyper Markov laws in the statistical analysis of decomposable graphical models. *The Annals of Statistics*, 21:1272–1317, 1993.

[3] M.D. Escobar and M. West. Bayesian density estimation and inference using mixtures. *Journal of the American Statistical Association*, 90:577–588, 1995.

[4] P. Fearnhead. Particle filters for mixture models with an unknown number of components. *Statistics and Computing*, 14:11–21, 2004.

[5] T.L. Griffiths and Z. Ghahramani. Infinite latent feature models and the Indian buffet process. In *Advances in Neural Information Processing Systems*, 2006.

[6] D. Heinz. Building hyper dirichlet processes for graphical models. *Electonic Journal of Statistics*, 3:290–315, 2009.

[7] J.F.C. Kingman. Random partitions in population genetics. *Proceedings of the Royal Society of London*, 361:1–20, 1978.

[8] S.L. Lauritzen. *Graphical Models*. Oxford University Press, 1996.

[9] P. Müller, F. Quintana, and G. Rosner. A method for combining inference across related nonparametric Bayesian models. *Journal of the Royal Statistical Society B*, 66:735–749, 2004.

[10] R.M. Neal. Markov chain sampling methods for Dirichlet process mixture models. *Journal of Computational and Graphical Statistics*, 9:249–265, 2000.

[11] J. Pitman. Exchangeable and partially exchangeable random partitions. *Probability theory and related fields*, 102:145–158, 1995.

[12] Y.W. Teh, M.I. Jordan, M.J. Beal, and D.M. Blei. Hierarchical Dirichlet processes. *Journal of the American Statistical Association*, 101:1566–1581, 2006.

[13] S. Walker and P. Muliere. A bivariate Dirichlet process. *Statistics and Probability Letters*, 64:1–7, 2003.

[14] F. Wood and T.L. Griffiths. Particle filtering for nonparametric Bayesian matrix factorization. In *Advances in Neural Information Processing Systems*, 2007.

